# Effects of Stress and Genotype on Meta-parameter Dynamics in Reinforcement Learning

**Gediminas Lukšys**[1,2]
gediminas.luksys@epfl.ch

**Jérémie Knüsel**[1]
jeremie.knuesel@epfl.ch

**Denis Sheynikhovich**[1]
denis.sheynikhovich@epfl.ch

**Carmen Sandi**[2]
carmen.sandi@epfl.ch

**Wulfram Gerstner**[1]
wulfram.gerstner@epfl.ch

[1]Laboratory of Computational Neuroscience
[2]Laboratory of Behavioral Genetics
Ecole Polytechnique Fédérale de Lausanne
CH-1015, Switzerland

## Abstract

Stress and genetic background regulate different aspects of behavioral learning through the action of stress hormones and neuromodulators. In reinforcement learning (RL) models, meta-parameters such as learning rate, future reward discount factor, and exploitation-exploration factor, control learning dynamics and performance. They are hypothesized to be related to neuromodulatory levels in the brain. We found that many aspects of animal learning and performance can be described by simple RL models using *dynamic control of the meta-parameters*. To study the effects of stress and genotype, we carried out 5-hole-box light conditioning and Morris water maze experiments with C57BL/6 and DBA/2 mouse strains. The animals were exposed to different kinds of stress to evaluate its effects on immediate performance as well as on long-term memory. Then, we used RL models to simulate their behavior. For each experimental session, we estimated a set of model meta-parameters that produced the best fit between the model and the animal performance. The dynamics of several estimated meta-parameters were qualitatively similar for the two simulated experiments, and with statistically significant differences between different genetic strains and stress conditions.

## 1 Introduction

Animals choose their actions based on reward expectation and motivational drives. Different aspects of learning are known to be influenced by acute stress [1, 2, 3] and genetic background [4, 5]. Stress effects on learning depend on the stress type (*eg* task-specific or unspecific) and intensity, as well as on the learning paradigm (*eg* spatial/episodic vs. procedural learning) [3]. It is known that stress can affect short- and long-term memory by modulating plasticity through stress hormones and neuromodulators [1, 2, 3, 6]. However, there is no integrative model that would accurately predict and explain differential effects of acute stress. Although stress factors can be described in quantitative measures, their effects on learning, memory, and performance are strongly influenced by how an animal perceives it. The subjective experience can be influenced by emotional memories as well as by behavioral genetic traits such as anxiety, impulsivity, and novelty reactivity [4, 5, 7].

In the present study, behavioral experiments conducted on two different genetic strains of mice and under different stress conditions were combined with a modeling approach. In our models, behavioral performance as a function of time was described in the framework of temporal difference reinforcement learning (TDRL).

In TDRL models [8] a modeled animal, termed *agent*, can occupy various states and undertake actions in order to acquire rewards. The expected values of cumulative future reward (Q-values) are learned by observing immediate rewards delivered under different state-action combinations. Their update is controlled by certain meta-parameters such as learning rate, future reward discount factor, and memory decay/interference factor. The Q-values (together with the exploitation/exploration factor) determine what actions are more likely to be chosen when the animal is at a certain state, *ie* they represent the goal-oriented behavioral strategy learned by the agent. The activity of certain neuromodulators in the brain are thought to be associated with the role the meta-parameters play in the TDRL models. Besides dopamine (DA), whose levels are known to be related to the TD reward prediction error [9], serotonin (5-HT), noradrenaline (NA), and acetylcholine (ACh) were discussed in relation to TDRL meta-parameters [10]. Thus, the knowledge of the characteristic meta-parameter dynamics can give an insight into the putative neuromodulatory activities in the brain. Dynamic parameter estimation approaches, recently applied to behavioral data in the context of TDRL [11], could be used for this purpose.

In our study, we carried out 5-hole-box light conditioning and Morris water maze experiments with C57BL/6 and DBA/2 inbred mouse strains (referred to as C57 and DBA from now on), renown for their differences in anxiety, impulsivity, and spatial learning [4, 5, 12]. We exposed subgroups of animals to different kinds of stress (such as motivational stress or task-specific uncertainty) in order to evaluate its effects on immediate performance, and also tested their long-term memory after a break of 4-7 weeks. Then, we used TDRL models to describe the mouse behavior and established a number of performance measures that are relevant to task learning and memory (such as mean response times and latencies to platform) in order to compare the outcome of the model with the animal performance. Finally, for each experimental session we ran an optimization procedure to find a set of the meta-parameters, best fitting to the experimental data as quantified by the performance measures. This approach made it possible to relate the effects of stress and genotype to differences in the meta-parameter values, allowing us to make specific inferences about learning dynamics (generalized over two different experimental paradigms) and their neurobiological correlates.

## 2   Reinforcement learning model of animal behavior

In the TDRL framework [8] animal behavior is modelled as a sequence of actions. After an action is performed, the animal is in a new state where it can again choose from a set of possible actions. In certain states the animal is rewarded, and the goal of learning is to choose actions so as to maximize the expected future reward, or *Q-value*, formally defined as

$$Q(s_t, a_t) = E\left(\sum_{k=0}^{\infty} \gamma^k r_{t+k+1}|s_t, a_t\right) \ , \tag{1}$$

where $(s_t, a_t)$ is the state-action pair, $r_t$ is a reward received at time step $t$ and $0 < \gamma < 1$ is the *future reward discount factor* which controls to what extent the future rewards are taken into account. As soon as state $s_{t+1}$ is reached and a new action is selected, the estimate of the previous state's value $Q(s_t, a_t)$ is updated based on the reward prediction error $\delta_t$ [8]:

$$\delta_t = r_{t+1} + \gamma Q(s_{t+1}, a_{t+1}) - Q(s_t, a_t) \ , \tag{2}$$

$$Q(s_t, a_t) \leftarrow Q(s_t, a_t) + \alpha\delta_t \ , \tag{3}$$

where $\alpha$ is the *learning rate*. The action selection at each state is controlled by the *exploitation factor* $\beta$ such that actions with high Q-values are chosen more often if the $\beta$ is high, whereas random actions are chosen most of the time if the $\beta$ is close to zero. Meta-parameters $\alpha$, $\beta$ and $\gamma$ are the free parameters of the model.

## 3   5-hole-box experiment and modeling

Experimental subjects were male mice (24 of the C57 strain, and 24 of the DBA strain), 2.5-month old at the beginning of the experiment, and food deprived to 85-90% of the initial weight. During an

experimental session, each animal was placed into the 5-hole-box (5HB) (Figure 1a). The animals had to learn to make a nose poke into any of the holes upon the onset of lights and not to make it in the absence of light. After the response to light, the animals received a reward in form of a food pellet. Once a poke was initiated (see *starting a poke* in Figure 1b), the mouse had to stay in the hole at least for a short time (0.3-0.5 sec) in order to find the delivered reward (*continuing a poke*). Trial ended (lights turned off) as soon as the nose poke was finished. If the mouse did not find the reward, the reward remained in the box and the animal could find it during the next poke in the same box. The inter-trial interval (ITI) between subsequent trials was 15 sec. However, a new trial could only start when during the last 3 sec before it there were no wrong (ITI) pokes, so as to penalize spontaneous poking. The total session time was 10 min. Hence, the number of trials depended on how fast animals responded to light and how often they made ITI pokes.

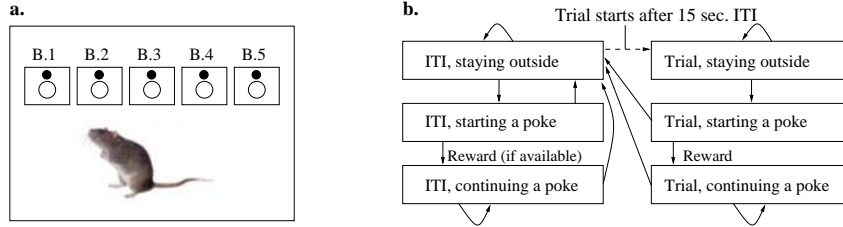

Figure 1: **a.** Scheme of the 5HB experiment. Open circles are the holes where the food is delivered, filled circles are the lights. All 5 holes were treated as equivalent during the experiment. **b.** 5HB state-action chart. Rectangles are states, arrows are actions.

After 2 days of habituation, during which the mice learned that food could be delivered in the holes, they underwent 8 consecutive days of training. During days 5-7 subsets of the animals were exposed to different stress conditions: motivational stress (MS, food deprivation to 85-87% of the initial weight vs. 88-90% in controls) and uncertainty in the reward delivery (US, in 50% of correct responses they received either none or 2 food pellets). Mice of each strain were divided into 4 stress groups: controls, MS, US, and MS+US. After a break of 26 days the long-term memory of the mice was tested by retraining them for another 8 days. During days 5-8 of the retraining, we again evaluated the impact of stress factors by exposing half of the mice to extrinsic stress (ES, 30 min on an elevated platform right before the 5HB experiment).

To model the mouse behavior we used a discrete state TDRL model with 6 states: [*ITI*, *trial*] × [*staying outside*, *starting a poke*, *continuing a poke*], and 2 actions: *move* (in or out), and *stay* (see Figure 1b). Actions were chosen according to the soft-max method [8]:

$$p(a|s) = \exp(\beta Q(s, a)) / \sum_k \exp(\beta Q(s, a_k)) \ , \qquad (4)$$

where $k$ runs over all actions and $\beta$ is the exploitation factor. Initial Q-values were equal to zero. Since the time spent outside the holes was comparatively long and included multiple (task irrelevant) actions, state/action pair *staying outside*/*stay* was given much more weight in the above formula. The time step (0.43 sec) was constant throughout the experiment and was chosen to fit the animal performance in the *beginning* of the experiment. Finally, to account for the memory decay after each day all $Q(s, a)$ values were updated as follows:

$$Q(s, a) \leftarrow Q(s, a) \cdot (1 - \lambda) + \langle Q(s, a) \rangle_{s,a} \cdot \lambda \ , \qquad (5)$$

where $\lambda$ is a memory decay/interference factor, and $\langle Q(s, a) \rangle_{s,a}$ is the average over Q values for all states and all actions at the end of the day.

All performance measures (PMs) used in the 5HB paradigm (number of trials, number of ITI pokes, mean response time, mean poke length, $\mathrm{TimePref}$[1] and $\mathrm{LengthPref}$[2]) were evaluated over the entire session (10 min, 1400 time steps), during which different states[3] could be visited multiple

times. As opposed to an online "SARSA"-type update of Q-values, we work with state occupancy probabilities $p(s_t)$ and update Q-values with the following reward prediction error:

$$\delta_t = \mathrm{E}[r_t] - Q(a_t, s_t) + \gamma \sum_{\forall a_{t+1}, s_{t+1}} Q(a_{t+1}, s_{t+1}) \cdot p(a_{t+1}, s_{t+1} | a_t, s_t) \ . \tag{6}$$

## 4   Morris water maze experiment and modeling

The same mice as in the 5HB (4.5-month old at the beginning of the experiment) were tested in a variant of the Morris water maze (WM) task [13]. Starting from one of 4 starting positions in the circular pool filled with an opaque liquid they had to learn the location of a hidden escape platform using stable extra-maze cues (Fig. 2a). Animals were initially trained for 4 days with 4 sessions a day (to avoid confusion with 5HB, we consider each WM session consisting of only one trial). Trial length was limited to 60s, and the inter-session interval was 25 min.). Half of the mice had to swim in cold water of 19°C (motivational stress, MS), while the rest were learning at 26°C (control).

After a 7-week break, 3-day long memory testing was done at 22-23°C for all animals. Finally, after another 2 weeks, the mice performed the task for 5 more days: half of them did a version with uncertainty stress (US), where the platform location was randomly varying between the old position and its rotationally opposite; the other half did the same task as before.

Behavior was quantified using the following 4 PMs: time to reach the goal (escape latency), time spent in the target platform quadrant, the opposite platform quadrant, and in the wall region (Fig. 2a).

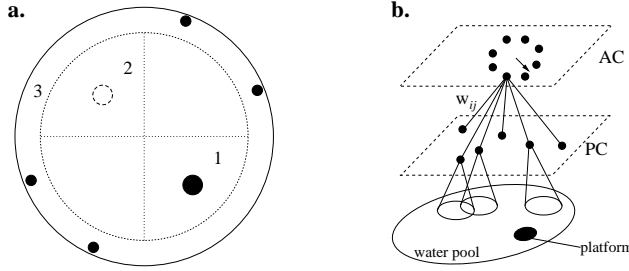

Figure 2: WM experiment and model. **a.** Experimental setup. *1* – target platform quadrant, *2* – opposite platform quadrant, *3* – wall region. Small filled circles mark 4 starting positions, large filled circle marks the target platform, open circle marks the opposite platform (used only in the US condition), pool $\varnothing = 1.4m$. **b.** Activities of place cells (PC) encode position of the animal in the WM, activities of action cells encode direction of the next movement.

A TDRL paradigm (1)-(3) in continuous state and action spaces has been used to model the mouse behavior in the WM [14, 15]. The position of the animal is represented as a population activity of $N_{\mathrm{pc}} = 211$ 'place cells' (PC) whose preferred locations are distributed uniformly over the area of a modelled circular arena (Fig. 2b). Activity of place cell $j$ is modelled by a Gaussian centered at the preferred location $\vec{p}_j$ of the cell:

$$r_j^{\mathrm{pc}} = \exp(-\|\vec{p} - \vec{p}_j\|^2 / 2\sigma_{\mathrm{pc}}^2) \ , \tag{7}$$

where $\vec{p}$ is the current position of the modelled animal and $\sigma_{\mathrm{pc}} = 0.25$ defines the width of the spatial receptive field relative to the pool radius. Place cells project to the population of $N_{\mathrm{ac}} = 36$ 'action cells' (AC) via feed-forward all-to-all connections with modifiable weights. Each action cell is associated with angle $\phi_i$, all $\phi_i$ being distributed uniformly in $[0, 2\pi]$. Thus, an activity profile on the level of place cells (i.e. state $s_t$) causes a different activity profile on the level of the action cells depending on the value of the weight vector. The activity of action cell $i$ is considered as the *value* of the action (defined as a movement in direction $\phi_i$[4]):

$$Q(s_t, a_t) = r_i^{\mathrm{ac}} = \sum_j w_{ij} r_j^{\mathrm{pc}} \ . \tag{8}$$

The action selection follows $\epsilon$-greedy policy, where the optimal action $a^*$ is chosen with probability $\beta = 1 - \epsilon$ and a random action with probability $1 - \beta$. Action $a^*$ is defined as movement in the direction of the center of mass $\phi^*$ of the AC population[5]. Q-value corresponding to an action with continuous angle $\phi$ is calculated as linear interpolation between activities of the two closest action cells. During learning the PC→AC connection weights are updated on each time step in such a way as to decrease the reward prediction error $\delta_t$ (3):

$$\Delta w_{ij} = \alpha \delta r_i^{\text{ac}} r_j^{\text{pc}} \ . \tag{9}$$

The Hebbian-like form of the update rule (9) is due to the fact that we use distributed representations for states and actions, i.e. there is no single state/action pair responsible for the last movement.

To simulate one experimental session it is necessary to (*i*) initialize the weight matrix $\{w_{ij}\}$, (*ii*) choose meta-parameter values and starting position $\vec{p}_0$, (*iii*) compute (7)-(8) and perform corresponding movements until $\|\vec{p} - \vec{p}_{\text{pl}}\| < R_{\text{pl}}$ at which point reward $r = 15$ is delivered ($R_{\text{pl}}$ is the platform radius). Wall hits result in a small negative reward ($r_{\text{wall}} = -3$).

For each session and each set of the meta-parameters, 48 different sets of random initial weights $w_{ij}$ (corresponding to individual mice) were used to run the model, with 50 simulations started out of each set. Final values of the PMs were averaged over all repetitions for each subgroup of mice.

To account for the loss of memory, after each day all weights were updated as follows:

$$w_{ij}^{\text{new}} = w_{ij}^{\text{old}} \cdot (1 - \lambda) + w_{ij}^{\text{initial}} \cdot \lambda \tag{10}$$

where $\lambda$ is the memory decay factor, $w_{ij}^{\text{old}}$ is the weight value at the end of the day, and $w_{ij}^{\text{initial}}$ is the initial weight value before any learning took place.

## 5  Goodness-of-fit function and optimization procedure

To compare the model with the experiment we used the following goodness-of-fit function [16]:

$$\chi^2 = \sum_{k=1}^{N_{\text{PM}}} (\text{PM}_k^{\text{exp}} - \text{PM}_k^{\text{mod}}(\alpha, \beta, \gamma, \lambda))^2 / (\sigma_k^{\text{exp}})^2 \ , \tag{11}$$

where $\text{PM}_k^{\text{exp}}$ and $\text{PM}_k^{\text{mod}}$ are the PMs calculated for the animals and the model, respectively and $N_{\text{PM}}$ is the number of the PMs. $\text{PM}_k^{\text{mod}}(\alpha, \beta, \gamma, \lambda)$ are calculated after simulation of one session with fixed values of the meta-parameters. $\text{PM}_k^{\text{exp}}$ were calculated either for each animal (5HB), or for each subgroup (WM). Using stochastic gradient ascent, we minimized (11) with respect to $\alpha, \beta, \gamma$ for each session separately by systematically varying the meta-parameters in the following ranges: for WM, $\alpha \in [10^{-5}, 5 \cdot 10^{-2}]$ and $\beta, \gamma \in [0.01, 0.99]$, and for 5HB, $\alpha, \gamma \in [0.03, 0.99]$ and $\beta \in [0.3, 9.9]$. Decay factor $\lambda \in [0.01, 0.99]$ was estimated only for the first session after the break, otherwise constant values of $\lambda = 0.03$ (5HB) and $\lambda = 0.2$ (WM) were used.

Several control procedures were performed to ensure that the meta-parameter optimization was statistically efficient and self-consistent. To evaluate how well the model fits the experimental data we used $\chi^2$-test with $\nu = N_{\text{PM}} - 3$ degrees of freedom (since most of the time we had only 3 free meta-parameters). The $P(\chi^2, \nu)$ value, defined as the probability that a realization of a chi-square-distributed random variable would exceed $\chi^2$ by chance, was calculated for each session separately. Generally, values of $P(\chi^2, \nu) > 0.01$ correspond to a fairly good model [16]. To check reliability of the estimated meta-parameters we used the same optimization procedure with $\text{PM}_k^{\text{exp}}$ artificially generated by the model itself. In a self-consistent model such a procedure is expected to find meta-parameter values similar to those with which the PMs were generated. Finally, to see how well the model generalizes to previously unseen data, we used half of the available experimental data for optimization and tested the estimated parameters on the other half. Then we evaluated $\chi^2$ and $P(\chi^2, \nu)$ values for the testing as well as the training data.

## 6  Results

The meta-parameter estimation procedure was performed for the models of both experiments using stochastic gradient ascent in $\chi^2$ goodness-of-fit. For the 5HB, meta-parameters were estimated for

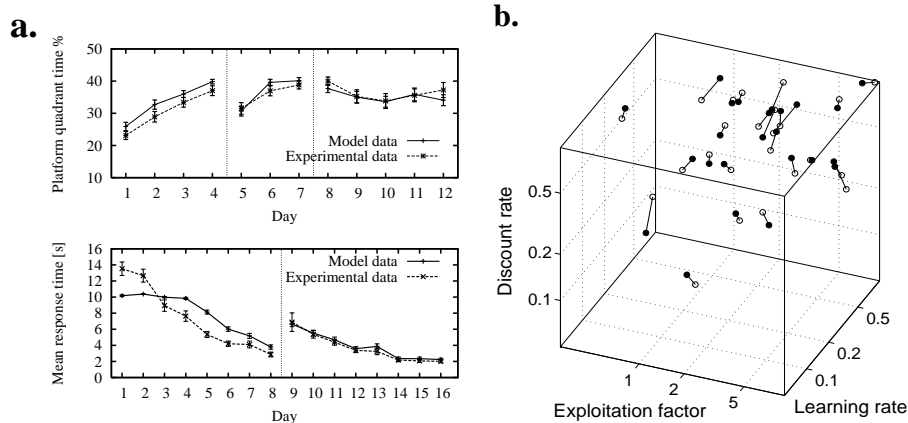

Figure 3: **a.** Example of PM evolution with learning in the WM (platform quadrant time, top) and in the 5HB (mean response time, bottom). **b.** Self-consistency check: true (open circles) and estimated (filled circles) meta-parameter values for the 24 random sets in the 5HB

each animal and each experimental day. Further (sub)group values were calculated by averaging the individual estimations. For the WM, meta-parameters were estimated for each subgroup and each experimental session. Learning dynamics in both experiments are illustrated in Figure 3a for 2 representative PMs, where average performances for all mice and the corresponding models (with estimated meta-parameters) are shown.

The results of both meta-parameter estimation procedures indicated a reasonably good fit between the model and animal performance. Evaluating the testing data, the condition $P(\chi^2, \nu) > 0.01$ was satisfied for 92.5% of 5HB estimated parameter sets, and for 98.4% in the WM. The mean $\chi^2$ values for the testing data were $\langle \chi^2 \rangle = 1.59$ in the WM ($P(\chi^2, 1) = 0.21$) and $\langle \chi^2 \rangle = 5.27$ in the 5HB ($P(\chi^2, 3) = 0.15$). There was a slight over-fitting only in the WM estimation.

To evaluate the quality of the estimated optima and sensitivities to different meta-parameters, we calculated eigenvalues of the Hessian of $1/\chi^2$ around each of the estimated points. 98.4% of all eigenvalues were negative, and most of the corresponding eigenvectors were aligned with the directions of $\alpha$, $\beta$, and $\gamma$, indicating that there were no significant correlations in parameter estimation. Furthermore, the absolute eigenvalues were highest in the directions of $\beta$ and $\gamma$, thus the error surface is steep along these meta-parameters. To test the reliability of estimated meta-parameters, the self-consistency check was performed using a number of random meta-parameter sets. The mean absolute errors (distances between real and estimated parameter values) were quite small for exploitation factors ($\beta$) – approximately 6% of the total range, but higher for the reward discount factors ($\gamma$) and for the learning rates ($\alpha$) – 10-29% of the total range (Figure 3b). This indicates that estimated $\beta$ values should be considered more reliable than those of $\alpha$ and $\gamma$.

## 6.1 Meta-parameter dynamics

During the course of learning, exploitation factors ($\beta$) (Figure 4a,b) showed progressive increase (regression $p \ll 0.001$ for both the 5HB and the WM), reaching the peak at the end of each learning block. They were consistently higher for the C57 mice than for the DBA mice (2-way ANOVA with replications, $p \ll 0.001$ for both experiments), indicating that the DBA mice were exploring the environment more actively, and/or were not able to focus their attention well on the specific task. Finally, C57 mouse groups, exposed to motivational stress in the WM and to extrinsic stress in the 5HB, had elevated exploitation factors (ANOVA $p < 0.01$ for both experiments), however there was no effect for the DBA mice.

The estimated learning rates ($\alpha$) did not show any obvious changes or trends with learning for either 5HB or WM. There were no differences between the 2 genetic strains (nor among the stress conditions) with one exception: for the first several days of the training, C57 learning rates were

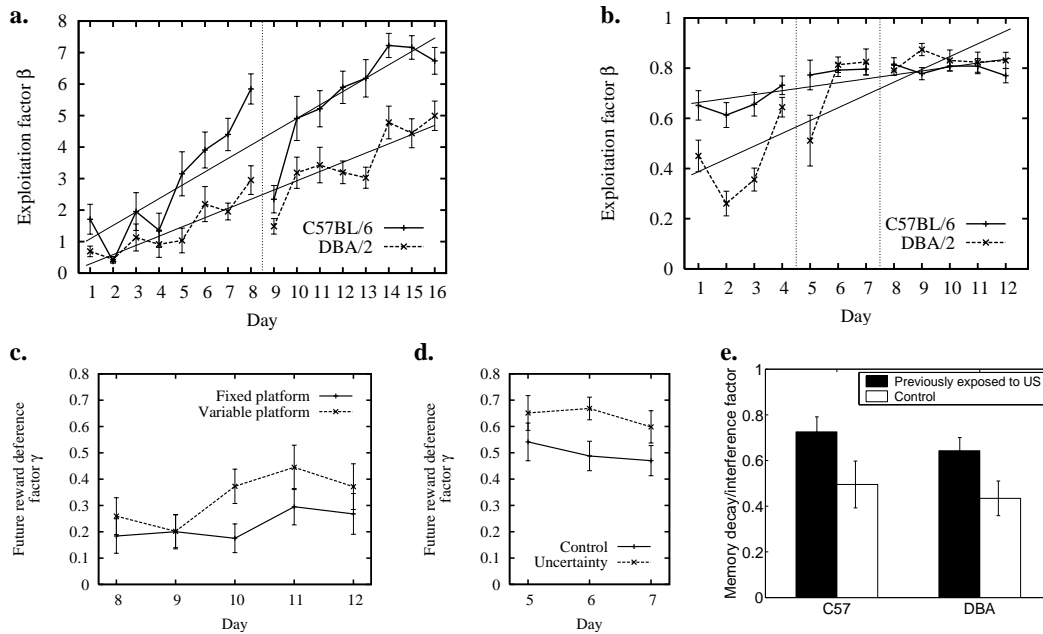

Figure 4: **a,b.** Estimated exploitation factors $\beta$ for 5HB (**a**, break is between days 8 & 9) and WM (**b**, breaks between days 4 & 5 and between 7 & 8). **c,d.** Estimated future reward deference factors for the variable platform trials in the WM (**c**) and for the uncertainty trials in the 5HB (**d**). **e.** Estimated memory decay / interference factors for the first day after the break in the 5HB.

significantly higher (ANOVA $p < 0.01$ in both experiments), indicating that C57 mice could learn a novel task more quickly.

Under uncertainty (in reward delivery for the 5HB, and in the target platform location for the WM) future reward discount factors ($\gamma$) were significantly elevated (ANOVA $p < 0.02$, Figure 4c,d). In the 5HB, memory decay factors ($\lambda$), estimated for the first day after the break, were significantly higher ($p < 0.01$, unpaired t-test) for animals, previously exposed to uncertainty (Figure 4e). This suggests that uncertainty makes animals consider rewards further into the future, and it seems to impair memory consolidation.

## 7 Discussion

In this paper we showed that various behavioral outcomes (caused by genetic traits and/or stress factors) could be predicted by our TDRL models for 2 different tasks. This provides hypotheses concerning the neuromodulatory mechanisms, which we plan to test using pharmacological manipulations (typically, injections of agonists or antagonists of relevant neurotransmitter systems).

Results for the exploitation factors suggest that with learning (and decreasing reward prediction errors) the acquired knowledge is used more for choosing actions. This might also be related to decreased subjective stress and higher stressor controllability. The difference between C57 and DBA strains shows two things. Firstly, the anxious DBA mice cannot exploit their knowledge as well as C57 can. Secondly, in response to motivational or extrinsic stress C57 mice are the only ones that increase their exploitation. This may be related to an inverse-U-shaped effect of the noradrenergic influences on focused attention and performance accuracy [17]. Animals with low anxiety (C57) might be on the left side of the curve, and additional stress might lead them to optimal performance, while those with high anxiety – already on the right side, leading to possibly impaired performance. Our results may also suggest that the widely proclaimed deficiency of DBA mice in spatial learning (as compared to C57) [4, 12] might be primarily due to differential attentional capabilities.

The increased future reward discount factors under uncertainty indicate a reasonable adaptive response – animals should not concentrate their learning on immediate events when task-reward rela-

tions become ambiguous. Uncertainty in behaviorally relevant outcomes under stress causes a decrease in subjective stressor controllability, which is known to be related to elevated serotonin levels [18]. Higher memory decay / interference factors for the animals previously exposed to uncertainty could be due to partially impaired memory consolidation and/or due to stronger competition between different strategies and perceptions of the uncertain task.

Although estimated meta-parameter values can be easily compared between certain experimental conditions, it is difficult to study in this way the interactions between different genetic and environmental factors or extrapolate beyond the limits of available conditions. One could overcome this disadvantage by developing a black-box parameter model that would help us to evaluate in a flexible way the contributions of specific factors (motivation, uncertainty, genotype) to meta-parameter dynamics, as well as their relationship with dynamics of TD errors ($\delta_t$) during the process of learning.

**Acknowledgments**

This work was partially supported by a grant from the Swiss National Science Foundation to C.S. (3100A0-108102).

## Footnotes

[1]$\mathrm{TimePref}$ = (average time between adjacent ITI pokes) / (average response time)

[2]$\mathrm{LengthPref}$ = (average response length) / (average ITI poke length)

[3]including the pseudo-states, corresponding to time steps within the 15 sec ITI

[4]A constant step length was chosen to fit the average speed of the animals during the experiment

[5]i.e. $\phi^* = \arctan(\sum_i r_i^{\text{ac}} \sin(2\pi k/N_{\text{ac}}) / \sum_i r_i^{\text{ac}} \cos(2\pi k/N_{\text{ac}}))$

# References

[1] J. J. Kim and D. M. Diamond. The stressed hippocampus, synaptic plasticity and lost memories. *Nat Rev Neurosci.*, 3(6):453–62., Jun 2002.

[2] C. Sandi, M. Loscertales, and C. Guaza. Experience-dependent facilitating effect of corticosterone on spatial memory formation in the water maze. *Eur J Neurosci.*, 9(4):637–42., Apr 1997.

[3] M. Joels, Z. Pu, O. Wiegert, M. S. Oitzl, and H. J. Krugers. Learning under stress: how does it work? *Trends Cogn Sci.*, 10(4):152–8. Epub 2006 Mar 2., Apr 2006.

[4] J. M. Wehner, R. A. Radcliffe, and B. J. Bowers. Quantitative genetics and mouse behavior. *Annu Rev Neurosci.*, 24:845–67., 2001.

[5] A. Holmes, C. C. Wrenn, A. P. Harris, K. E. Thayer, and J. N. Crawley. Behavioral profiles of inbred strains on novel olfactory, spatial and emotional tests for reference memory in mice. *Genes Brain Behav.*, 1(1):55–69., Jan 2002.

[6] J. L. McGaugh. The amygdala modulates the consolidation of memories of emotionally arousing experiences. *Annu Rev Neurosci.*, 27:1–28., 2004.

[7] M. J. Kreek, D. A. Nielsen, E. R. Butelman, and K. S. LaForge. Genetic influences on impulsivity, risk taking, stress responsivity and vulnerability to drug abuse and addiction. *Nat Neurosci.*, 8:1450–7, 2005.

[8] R. Sutton and A. G. Barto. *Reinforcement Learning - An Introduction*. MIT Press, 1998.

[9] W. Schultz, P. Dayan, and P. R. Montague. A neural substrate of prediction and reward. *Science*, 275(5306):1593–9, Mar 14 1997.

[10] K. Doya. Metalearning and neuromodulation. *Neural Netw*, 15(4-6):495–506, Jun-Jul 2002.

[11] K. Samejima, K. Doya, Y. Ueda, and M. Kimura. Estimating internal variables and paramters of a learning agent by a particle filter. In *Advances in Neural Information Processing Systems 16*. 2004.

[12] C. Rossi-Arnaud and M. Ammassari-Teule. What do comparative studies of inbred mice add to current investigations on the neural basis of spatial behaviors? *Exp Brain Res.*, 123(1-2):36–44., Nov 1998.

[13] R. G. M. Morris. Spatial localization does not require the presence of local cues. *Learning and Motivation*, 12:239–260, 1981.

[14] D. J. Foster, R. G. M. Morris, and P. Dayan. A model of hippocampally dependent navigation, using the temporal difference learning rule. *Hippocampus*, 10(1):1–16, 2000.

[15] T. Strösslin, D. Sheynikhovich, R. Chavarriaga, and W. Gerstner. Modelling robust self-localisation and navigation using hippocampal place cells. *Neural Networks*, 18(9):1125–1140, 2005.

[16] W. H. Press, B. P. Flannery, S. A. Teukolsky, and W. T. Vetterling. *Numerical Recipes in C : The Art of Scientific Computing*. Cambridge University Press, 1992.

[17] G. Aston-Jones, J. Rajkowski, and J. Cohen. Locus coeruleus and regulation of behavioral flexibility and attention. *Prog Brain Res.*, 126:165–82., 2000.

[18] J. Amat, M. V. Baratta, E. Paul, S. T. Bland, L. R. Watkins, and S. F. Maier. Medial prefrontal cortex determines how stressor controllability affects behavior and dorsal raphe nucleus. *Nat Neurosci.*, 8(3):365–71. Epub 2005 Feb 6., Mar 2005.
